# Reading a Neural Code

**William Bialek, Fred Rieke, R. R. de Ruyter van Steveninck[1] and David Warland**
Department of Physics, and
Department of Molecular and Cell Biology
University of California at Berkeley
Berkeley, California 94720

## ABSTRACT

Traditional methods of studying neural coding characterize the encoding of known stimuli in average neural responses. Organisms face nearly the opposite task — decoding short segments of a spike train to extract information about an unknown, time-varying stimulus. Here we present strategies for characterizing the neural code from the point of view of the organism, culminating in algorithms for real-time stimulus reconstruction based on a single sample of the spike train. These methods are applied to the design and analysis of experiments on an identified movement-sensitive neuron in the fly visual system. As far as we know this is the first instance in which a direct "reading" of the neural code has been accomplished.

## 1  Introduction

Sensory systems receive information at extremely high rates, and much of this information must be processed in real time. To understand real-time signal processing in biological systems we must understand the representation of this information in neural spike trains. We ask several questions in particular:

- Does a single neuron signal only the occurrence of particular stimulus "features," or can the spike train represent a continuous time-varying input?

- How much information is carried by the spike train of a single neuron?

- Is the reliability of the encoded signal limited by noise at the sensory input or by noise and inefficiencies in the subsequent layers of neural processing?

- Is the neural code robust to errors in spike timing, or do realistic levels of synaptic noise place significant limits on information transmission?

- Do simple analog computations on the encoded signals correspond to simple manipulations of the spike trains?

Although neural coding has been studied for more than fifty years, clear experimental answers to these questions have been elusive (Perkel & Bullock, 1968; de Ruyter van Steveninck & Bialek, 1988). Here we present a new approach to the characterization of the neural code which provides explicit and sometimes surprising answers to these questions when applied to an identified movement-sensitive neuron in the fly visual system.

We approach the study of spiking neurons from the point of view of the organism, which, based only on the spike train, must estimate properties of an unknown time-varying stimulus. Specifically we try to solve the problem of *decoding* the spike train to recover the stimulus in real time. As far as we know our work is the first instance in which it has been possible to "read" the neural code in this literal sense. Once we can read the code, we can address the questions posed above. In this paper we focus on the code reading algorithm, briefly summarizing the results which follow.

## 2   Theoretical background

The traditional approach to the study of neural coding characterizes the *encoding* process: For an arbitrary stimulus waveform $s(\tau)$, what can we predict about the spike train? This process is completely specified by the conditional probability distribution $P[\{t_i\}|s(\tau)]$ of the spike arrival times $\{t_i\}$ conditional on the stimulus $s(\tau)$. In practice one cannot characterize this distribution in its entirety; most experiments result in only the lowest moment — the firing rate as function of time given the stimulus.

The classic experiments of Adrian and others established that, for static stimuli, the resulting constant firing rate provides a measure of stimulus strength. This concept is easily extended to any stimulus waveform which is characterized by constant parameters, such as a single frequency or fixed amplitude sine wave. Much of the effort in studying the encoding of sensory signals in the nervous system thus reduces to probing the relation between these stimulus parameters and the resulting firing rate. Generalizations to time-varying firing rates, especially in response to periodic signals, have also been explored.

The firing rate is a continuous function of time which measures the probability per unit time that the cell will generate a spike. The rate is thus by definition an average quantity; it is not a property of a single spike train. The rate can be estimated, in principle, by averaging over a large ensemble of redundant cells,

or by averaging responses of a single cell over repeated presentations of the same stimulus. This latter approach dominates the experimental study of spiking neurons. Measurements of firing rate rely on some form of redundancy — either the spatial redundancy of identical cells or the temporal redundancy of repeated stimuli. It is simply not clear that such redundancy exists in real sensory systems under natural stimulus conditions. In the absence of redundancy a characterization of neural responses in terms of firing rate is of little relevance to the signal processing problems faced by the organism. To say that "information is coded in firing rates" is of no use unless one can explain how the organism could estimate these firing rates by observing the spike trains of its own neurons.

We believe that none of the existing approaches[2] to neural coding addresses the basic problem of real-time signal processing with neural spike trains: The organism must extract information about continuously varying stimulus waveforms using only the discrete sequences of spikes. Real-time signal processing with neural spike trains thus involves some sort of interpolation between the spikes that allows the organism to estimate a continuous function of time.

The most basic problem of real-time signal processing is to *decode* the spike train and recover an estimate of the stimulus waveform itself. Clearly if we can accomplish this task then we can begin to understand how spike trains can be manipulated to perform more complex computations; we can also address the quantitative issues outlined in the Introduction. Because of the need to interpolate between spikes, such decoding is not a simple matter of inverting the conventional stimulus-response (rate) relations. In fact it is not obvious *a priori* that true decoding is even possible.

One approach to the decoding problem is to construct models of the encoding process, and proceed analytically to develop algorithms for decoding within the context of the model (Bialek & Zee, 1990). Using the results of this approach we can predict that linear filtering will, under some conditions, be an effective decoding algorithm, and we can determine the form of the filter itself. In this paper we have a more limited goal, namely to see if the *class* of decoding algorithms identified by Bialek and Zee is applicable to a real neuron. To this end we will treat the structure of the decoding filter as unknown, and find the "best" filter under given experimental conditions.

We imagine building a set of (generally non-linear) filters $\{F_n\}$ which operate on the spike train to produce an estimate of the stimulus. If the spikes arrive at times $\{t_i\}$, we write our estimate of the signal as a generalized convolution,

$$s_{\text{est}}(t) = \sum_i F_1(t - t_i) + \sum_{i,j} F_2(t - t_i, t - t_j) + \cdots. \tag{1}$$

How good are the reconstructions? We separate systematic and random errors by introducing a frequency dependent gain $g(\omega)$ such that $\langle|\tilde{s}(\omega)|\rangle = g(\omega)\langle|\tilde{s}_{est}(\omega)|\rangle$. The resulting gain is approximately unity through a reasonable bandwidth. Further, the distribution of deviations between the stimulus and reconstruction is approximately Gaussian. The absence of systematic errors suggests that non-linearities in the reconstruction filter are unlikely to help. Indeed, the contribution from the second order term in Eq. (1) to the reconstructions is negligible.

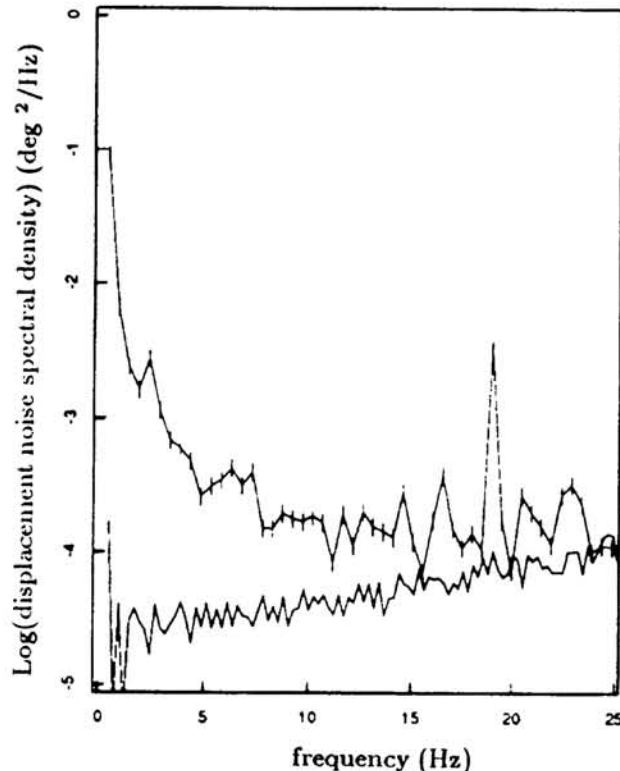

**Figure 2:** Spectral density of displacement noise from our reconstruction (upper curve). By multiplying the displacement noise level by a bandwidth, we obtain the square of the angular resolution of H1 for a step displacement. For a reasonable bandwidth the resolution is much less than the photoreceptor spacing, 1.35° — *"hyperacuity."* Also shown is the limit to the resolution of small displacements set by noise in the photoreceptor array (lower curve).

We identify the noise at frequency $\omega$ as the difference between the stimulus and the normalized reconstruction, $\tilde{n}(\omega) = \tilde{s}(\omega) - g(\omega)\tilde{s}_{est}(\omega)$. We then compute the spectral density (noise power per unit bandwidth) of the displacement noise (Fig 2). The noise level achieved in H1 is astonishing; with a one second integration time an observer of the spike train in H1 could judge the amplitude of a low frequency dither to 0.01° — more than one hundred times less than the photoreceptor spacing! If the fly's neural circuitry is noiseless, the fundamental limits to displacement resolution

stimulus,

$$F_1(\tau) = \int \frac{d\omega}{2\pi} e^{-i\omega\tau} \frac{\left\langle \tilde{s}(\omega) \sum_j e^{-i\omega t_j} \right\rangle}{\left\langle \sum_{i,j} e^{i\omega(t_i - t_j)} \right\rangle}. \tag{2}$$

The averages $\langle \cdots \rangle$ are with respect to an ensemble of stimuli $s(\tau)$.

2. Minimize $\chi^2$ with respect to purely causal functions. This may be done analytically, or numerically by expanding $F_1(\tau)$ in a complete set of functions which vanish at negative times, then minimizing $\chi^2$ by varying the coefficients of the expansion. In this method we must explicitly introduce a delay time which measures the lag between the true stimulus and our reconstruction.

We use the filter generated from the first method (which is the best possible linear filter) to check the filter generated by the second method. Fig. 1 illustrates reconstructions using these two methods. The filters themselves are also shown in the figure; we see that both methods give essentially the same answer.

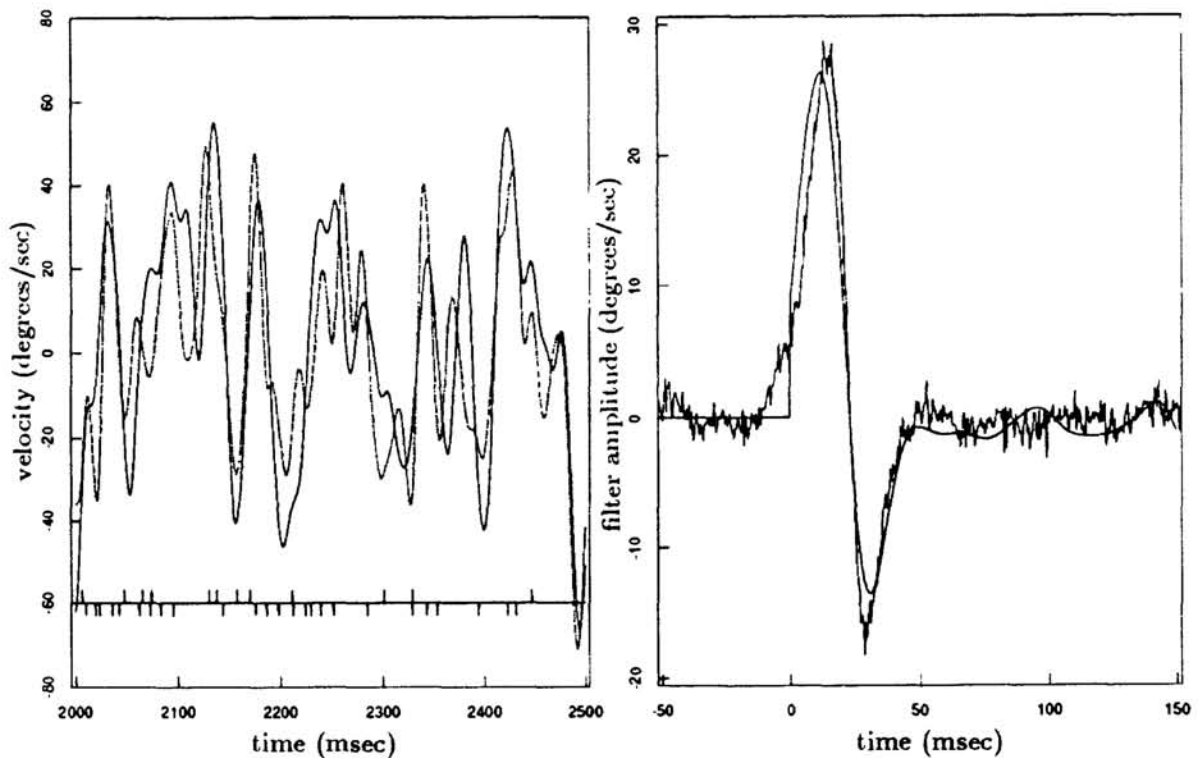

**Figure 1:** First order reconstruction $s_{est}(\tau)$ using method 1 (solid line). The stimulus is shown here as a dotted line for comparison. The reconstruction shown is for a segment of the spike train which was not used in the filter calculations. The spike train is shown at the bottom of the figure, where the negative spikes are from the "other eye" (cf. footnote 3). Both stimulus and reconstruction are smoothed with a 5 msec half-width Gaussian filter. The filters calculating using both methods are shown on the right.

We define the optimal filter to be that which minimizes $\chi^2 = \int dt |s(t) - s_{\text{est}}(t)|^2$, where $s(t)$ is the true stimulus, and the integration is over the duration of the experiment.

To insure that the filters we calculate allow real-time decoding, we require that the filters be causal, for example $F_1(\tau < 0) = 0$. But the occurrence of a spike at $t'$ conveys information about the stimulus at a time $t < t'$, so we must delay our estimate of the stimulus by some time $\tau_{delay} > t' - t$. In general we gain more information by increasing the delay, so we face a tradeoff: Longer waiting times allow us to gain more information but introduce longer reaction times to important stimuli. This tradeoff is exactly the tradeoff faced by the organism in reacting to external stimuli based on noisy and incomplete information.

## 3    Movement detection in the blowfly visual system

We apply our methods in experiments on a single wide field, movement-sensitive neuron (H1) in the visual system of the blowfly *Calliphora erythrocephela*. Flies and other insects exhibit visually guided flight; during chasing behavior course corrections can occur on time scales as short as 30 msec (Land & Collett, 1974). H1 appears to be an obligatory link in this control loop, encoding wide field horizontal movements (Hausen, 1984). Given that the maximum firing rate in H1 is 100-200 Hz, behavioral decisions must be based on the information carried by just a few spikes from this neuron. Further, the horizontal motion detection system consists of only a handful of neurons, so the fly has no opportunity to compute average responses (or firing rates).

In the experiments described here, the fly is looking at a rigidly moving random pattern (de Ruyter van Steveninck, 1986). The pattern is presented on an oscilloscope, and moved horizontally every 500 $\mu$sec in discrete steps chosen from an ensemble which approximates Gaussian white noise. This time scale is short enough that we can consider the resulting stimulus waveform $s(t)$ to be the instantaneous angular velocity. We record the spike arrival times $\{t_i\}$ extracellularly from the H1 neuron.[3]

## 4    First order reconstructions

To reconstruct the stimulus waveform requires that we find the filter $F_1$ which minimizes $\chi^2$. We do this in two different ways:

1. Disregard the constraint that the filter be causal. In this case we can write an explicit formula for the optimal filter in terms of the spike trains and the

are set by noise in the photoreceptor array. We have calculated these limits in the case where the displacements are small, which is true in our experiments at high frequencies. In comparing these limits with the results in H1 it is crucial that the photoreceptor signal and noise characteristics (de Ruyter van Steveninck, 1986) are measured under the same conditions as the H1 experiments analyzed here. It is clear from Fig. 2 that H1 approaches the theoretical limit to its performance. We emphasize that the noise spectrum in Fig. 2 is not a hypothetical measure of neural performance. Rather it is the real noise level achieved in our reconstructions. As far as we know this is the first instance in which the equivalent spectral noise level of a spiking neuron has been measured.

To explore the tradeoff between the quality and delay of the reconstruction we measure the cross-correlation of the smoothed stimulus with the reconstructions calculated using method 2 above for delays of 10-70 msec. For a delay of 10 msec the reconstruction carries essentially no information; this is expected since a delay of 10 msec is close to the intrinsic delay for phototransduction. As the delay is increased the reconstructions improve, and this improvement saturates for delays greater than 40 msec, close to the behavioral reaction time of 30 msec — the structure of the code is well matched to the behavioral decision task facing the organism.

## 5   Conclusions

Learning how to read the neural code has allowed us to quantify the information carried in the spike train independent of assumptions regarding the structure of the code. In addition, our analysis gives some hopefully more general insights into neural coding and computation:

*1. The continuously varying movement signal encoded in the firing of H1 can be reconstructed by an astonishingly simple linear filter.* If neurons summed their inputs and marked the crossing of thresholds (as in many popular models), such reconstructions would be impossible; the threshold crossings are massively ambiguous indicators of the signal waveform. We have carried out similar studies on a standard model neuron (the FitzHugh-Nagumo model), and find results similar to those in the H1 experiments. From the model neuron studies it appears that the linear representation of signals in spike trains is a general property of neurons, at least in a limited regime of their dynamics. In the near future we hope to investigate this statement in other sensory systems.

*2. The reconstruction is dominated by a "window" of ∼ 40 msec during which at most a few spikes are fired.* Because so few spikes are important, it does not make sense to talk about the "firing rate" — estimating the rate *vs.* time from observations of the spike train is at least as hard as estimating the stimulus itself!

*3. The quality of the reconstructions can be improved by accepting longer delays, but this improvement saturates at ∼ 30 − 40 msec, in good agreement with behavioral decision times.*

*4. Having decoded the neural signal we obtain a meaningful estimate of the noise level in the system and the information content of the code.* H1 accomplishes a real-time version of hyperacuity, corresponding to a noise level near the limits imposed by the quality of the sensory input. It appears that this system is close to achieving *optimal* real-time signal processing.

*5. From measurements of the fault tolerance of the code we can place requirements on the noise levels in neural circuits using the information coded in H1.* One of the standard objections to discussions of "spike timing" as a mechanism of coding is that there are no biologically plausible mechanisms which can make precise measurements of spike arrival times. We have tested the required timing precision by introducing timing errors into the spike train and characterizing the resulting reconstructions. Remarkably the code is "fault tolerant," the reconstructions degrading only slightly when we add timing errors of several msec.

Finally, we wish to emphasize our own surprise that it is so simple to recover time dependent signals from neural spike trains. The filters we have constructed are not very complicated, and they are linear. These results suggest that the representation of time-dependent sensory data in the nervous system is much simpler than we might have expected. We suggest that, correspondingly, simpler models of sensory signal processing may be appropriate.

# 6    Acknowledgments

We thank W. J. Bruno, M. Crair, L. Kruglyak, J. P. Miller, W. G. Owen, A. Zee, and G. Zweig for many helpful discussions. This work was supported by the National Science Foundation through a Presidential Young Investigator Award to WB, supplemented by funds from Cray Research and Sun Microsystems, and through a Graduate Fellowship to FR. DW was supported in part by the Systems and Integrative Biology Training Program of the National Institutes of Health. Initial work was supported by the Netherlands Organization for Pure Scientific Research (ZWO).

# 7    References

W. Bialek and A. Zee. *J. Stat. Phys.*, in press, 1990.

K. Hausen. In M. Ali, editor, *Photoreception and Vision in Invertebrates*. Plenum Press, New York and London, 1984.

M. Land and T. Collett. *J. Comp. Physiol.*, 89:331, 1974.

P. Marmarelis and V. Marmarelis. *Analysis of Physiological Systems. The White Noise Approach*. Plenum Press, New York, 1978.

D. Perkel and T. Bullock. *Neurosciences. Res. Prog. Bull.*, 6:221, 1968.

R. R. de Ruyter van Steveninck and W. Bialek. *Proc. R. Soc. Lond. B*, 234:379, 1988.

R. R. de Ruyter van Steveninck. *Real-time Performance of a Movement-sensitive Neuron in the Blowfly Visual System*. Rijksuniversiteit Groningen, Groningen, Netherlands, 1986.

## Footnotes

[1] Rijksuniversiteit Groningen, Postbus 30.001, 9700 RB Groningen The Netherlands

[2] Higher moments of the conditional probability $P[\{t_i\}|s(\tau)]$, such as the inter-spike interval distribution (Perkel & Bullock, 1968) are also average properties, not properties of single spike trains, and hence may not be relevant to real-time signal processing. White-noise methods (Marmarelis & Marmarelis, 1978) result in models which predict the time-varying firing rate in response to arbitrary input waveforms and thus suffer the same limitations as other rate-based approaches.

[3] There is one further caveat to the experiment. The firing rate in H1 is increased for back-to-front motion and is decreased for front-to-back motion; the dynamic range is much greater in the excitatory direction. The fly, however, achieves high sensitivity in both directions by combining information from both eyes. Because front-to-back motion in one eye corresponds to back-to-front motion in the other eye, we can simulate the two eye case while recording from only one H1 cell by using an antisymmetric stimulus waveform. We combine the information coded in the spike trains corresponding to the two "polarities" of the stimulus to obtain the information available from both H1 neurons.
